# Multi-stage Convex Relaxation for Learning with Sparse Regularization

**Tong Zhang**
Statistics Department
Rutgers University, NJ
tzhang@stat.rutgers.edu

## Abstract

We study learning formulations with non-convex regularizaton that are natural for sparse linear models. There are two approaches to this problem:

- Heuristic methods such as gradient descent that only find a local minimum. A drawback of this approach is the lack of theoretical guarantee showing that the local minimum gives a good solution.
- Convex relaxation such as $L_1$-regularization that solves the problem under some conditions. However it often leads to sub-optimal sparsity in reality.

This paper tries to remedy the above gap between theory and practice. In particular, we investigate a multi-stage convex relaxation scheme for solving problems with non-convex regularization. Theoretically, we analyze the behavior of a resulting two-stage relaxation scheme for the capped-$L_1$ regularization. Our performance bound shows that the procedure is superior to the standard $L_1$ convex relaxation for learning sparse targets. Experiments confirm the effectiveness of this method on some simulation and real data.

## 1 Introduction

Consider a set of input vectors $\mathbf{x}_1, \ldots, \mathbf{x}_n \in R^d$, with corresponding desired output variables $y_1, \ldots, y_n$. The task of supervised learning is to estimate the functional relationship $y \approx f(\mathbf{x})$ between the input $\mathbf{x}$ and the output variable $y$ from the training examples $\{(\mathbf{x}_1, y_1), \ldots, (\mathbf{x}_n, y_n)\}$. The quality of prediction is often measured through a loss function $\phi(f(\mathbf{x}), y)$. We assume that $\phi(f, y)$ is convex in $f$ throughout the paper. In this paper, we consider linear prediction model $f(\mathbf{x}) = \mathbf{w}^T \mathbf{x}$. As in boosting or kernel methods, nonlinearity can be introduced by including non-linear features in $\mathbf{x}$.

We are mainly interested in the scenario that $d \gg n$. That is, there are many more features than the number of samples. In this case, an unconstrained empirical risk minimization is inadequate because the solution overfits the data. The standard remedy for this problem is to impose a constraint on $\mathbf{w}$ to obtain a *regularized* problem. An important target constraint is *sparsity*, which corresponds to the (non-convex) $L_0$ regularization, defined as $\|\mathbf{w}\|_0 = |\{j : \mathbf{w}_j \neq 0\}| = k$. If we know the sparsity parameter $k$ for the target vector, then a good learning method is $L_0$ regularization:

$$\hat{\mathbf{w}} = \arg \min_{\mathbf{w} \in R^d} \frac{1}{n} \sum_{i=1}^{n} \phi(\mathbf{w}^T \mathbf{x}_i, y_i) \quad \text{subject to } \|\mathbf{w}\|_0 \leq k. \qquad (1)$$

If $k$ is not known, then one may regard $k$ as a tuning parameter, which can be selected through cross-validation. This method is often referred to as *subset selection* in the literature. Sparse learning is an essential topic in machine learning, which has attracted considerable interests recently. It can be shown that the solution of the $L_0$ regularization problem in (1) achieves good prediction accuracy

if the target function can be approximated by a sparse $\bar{\mathbf{w}}$. However, a fundamental difficulty with this method is the computational cost, because the number of subsets of $\{1, \ldots, d\}$ of cardinality $k$ (corresponding to the nonzero components of $\mathbf{w}$) is exponential in $k$.

Due to the computational difficult, in practice, it is necessary to replace (1) by some easier to solve formulations below:

$$\hat{\mathbf{w}} = \arg \min_{\mathbf{w} \in R^d} \frac{1}{n} \sum_{i=1}^{n} \phi(\mathbf{w}^T \mathbf{x}_i, y_i) + \lambda g(\mathbf{w}), \tag{2}$$

where $\lambda > 0$ is an appropriately chosen regularization condition. We obtain a formulation equivalent to (2) by choosing the regularization function as $g(\mathbf{w}) = \|\mathbf{w}\|_0$. However, this function is discontinuous. For computational reasons, it is helpful to consider a continuous approximation with $g(\mathbf{w}) = \|\mathbf{w}\|_p$, where $p > 0$. If $p \geq 1$, the resulting formulation is convex. In particular, by choosing the closest approximation with $p = 1$, one obtain *Lasso*, which is the standard convex relaxation formulation for sparse learning. With $p \in (0, 1)$, the $L_p$ regularization $\|\mathbf{w}\|_p$ is non-convex but continuous. In this paper, we are also interested in the following *capped-$L_1$* approximation of $\|\mathbf{w}\|_0$, with $g(\mathbf{w}) = \sum_{j=1}^{d} \min(|\mathbf{w}_j|, \alpha)$, where for $v \in R$: This is a good approximation to $L_0$ because as $\alpha \to 0$, $\sum_j \min(|\mathbf{w}_j|, \alpha)/\alpha \to \|\mathbf{w}\|_0$. Therefore when $\alpha \to 0$, this regularization condition is equivalent to the sparse $L_0$ regularization upto a rescaling of $\lambda$. Note that the capped-$L_1$ regularization is also non-convex. It is related to the so-called SCAD regularization in statistics, which is a smoother version. We use the simpler capped-$L_1$ regularization because the extra smoothness does not affect our algorithm or theory.

For a non-convex but smooth regularization condition such as capped-$L_1$ or $L_p$ with $p \in (0, 1)$, standard numerical techniques such as gradient descent leads to a local minimum solution. Unfortunately, it is difficult to find the global optimum, and it is also difficult to analyze the quality of the local minimum. Although in practice, such a local minimum solution may outperform the Lasso solution, the lack of theoretical (and practical) performance guarantee prevents the more wide-spread applications of such algorithms. As a matter of fact, results with non-convex regularization are difficult to reproduce because different numerical optimization procedures can lead to different local minima. Therefore the quality of the solution heavily depend on the numerical procedure used.

The situation is very difficult for a convex relaxation formulation such as $L_1$-regularization (Lasso). The global optimum can be easily computed using standard convex programming techniques. It is known that in practice, 1-norm regularization often leads to sparse solutions (although often suboptimal). Moreover, its performance has been theoretically analyzed recently. For example, it is known from the compressed sensing literature that under certain conditions, the solution of $L_1$ relaxation may be equivalent to $L_0$ regularization asymptotically even when noise is present (e.g. [3] and references therein). If the target is truly sparse, then it was shown in [9] that under some restrictive conditions referred to as *irrepresentable conditions*, 1-norm regularization solves the feature selection problem. The prediction performance of this method has been considered in [4, 8, 1].

Despite of its success, $L_1$-regularization often leads to suboptimal solutions because it is not a good approximation to $L_0$ regularization. Statistically, this means that even though it converges to the true sparse target when $n \to \infty$ (consistency), the rate of convergence can be suboptimal. The only way to fix this problem is to employ a non-convex regularization condition that is closer to $L_0$ regularization, such as the capped-$L_1$ regularization. The superiority of capped-$L_1$ is formally proved later in this paper.

Because of the above gap between practice and theory, it is important to study direct solutions of non-convex regularization beyond the standard $L_1$ relaxation. Our goal is to design a numerical procedure that leads to a *reproducible solution* with better theoretical behavior than $L_1$-regularization. This paper shows how this can be done. Specifically, we consider a general multi-stage convex relaxation method for solving learning formulations with non-convex regularization. In this scheme, concave duality is used to construct a sequence of convex relaxations that give better and better approximations to the original non-convex problem. Moreover, using the capped-$L_1$ regularization, we show that after only two stages, the solution gives better statistical performance than standard Lasso when the target is approximately sparse. In essence, this paper establishes a performance guarantee for non-convex formulations using a multi-stage convex relaxation approach that is more sophisticated than the standard one-stage convex relaxation (which is the standard approach com-

monly studied in the current literature). Experiments confirm the effectiveness of the multi-stage approach.

## 2   Concave Duality

Given a continuous regularization function $g(\mathbf{w})$ in (2) which may be non-convex, we are interested in rewriting it using concave duality. Let $\mathbf{h}(\mathbf{w}) : R^d \to \Omega \subset R^d$ be a map with range $\Omega$. It may not be a one-to-one map. However, we assume that there exists a function $\bar{g}_\mathbf{h}(\mathbf{u})$ defined on $\Omega$ such that $g(\mathbf{w}) = \bar{g}_\mathbf{h}(\mathbf{h}(\mathbf{w}))$ holds.

We assume that we can find $\mathbf{h}$ so that the function $\bar{g}_\mathbf{h}(\mathbf{u})$ is a concave function of $\mathbf{u}$ on $\Omega$. Under this assumption, we can rewrite the regularization function $g(\mathbf{w})$ as:

$$g(\mathbf{w}) = \inf_{\mathbf{v} \in R^d} \left[ \mathbf{v}^T \mathbf{h}(\mathbf{w}) + g_\mathbf{h}^*(\mathbf{v}) \right] \tag{3}$$

using concave duality [6]. In this case, $g_\mathbf{h}^*(\mathbf{v})$ is the concave dual of $\bar{g}_\mathbf{h}(\mathbf{u})$ given below

$$g_\mathbf{h}^*(\mathbf{v}) = \inf_{\mathbf{u} \in \Omega} \left[ -\mathbf{v}^T \mathbf{u} + \bar{g}_\mathbf{h}(\mathbf{u}) \right].$$

Moreover, it is well-known that the minimum of the right hand side of (3) is achieved at

$$\hat{\mathbf{v}} = \nabla_\mathbf{u} \bar{g}_\mathbf{h}(\mathbf{u})|_{\mathbf{u}=\mathbf{h}(\mathbf{w})}. \tag{4}$$

This is a very general framework. For illustration, we include two example non-convex sparse regularization conditions discussed in the introduction.

$L_p$ **regularization**  We consider the regularization condition $g(\mathbf{w}) = \sum_{j=1}^d |\mathbf{w}_j|^p$ for some $p \in (0, 1)$. Given any $q > p$, (3) holds with $\mathbf{h}(\mathbf{w}) = [|\mathbf{w}_1|^q, \ldots, |\mathbf{w}_d|^q]$ and $g_\mathbf{h}^*(\mathbf{v}) = c(p, q) \sum_j \mathbf{v}_j^{p/(p-q)}$ defined on the domain $\{\mathbf{v} : \mathbf{v}_j \geq 0\}$, where $c(p, q) = (q-p)p^{p/(q-p)}q^{q/(p-q)}$. In this case, $\bar{g}_\mathbf{h}(\mathbf{u}) = \sum_{j=1}^d \mathbf{u}_j^{p/q}$ on $\Omega = \{\mathbf{u} : \mathbf{u}_j \geq 0\}$. The solution in (4) is given by $\hat{\mathbf{v}}_j = (p/q)|\mathbf{w}_j|^{p-q}$.

**Capped-$L_1$ regularization**  We consider the regularization condition $g(\mathbf{w}) = \sum_{j=1}^d \min(|\mathbf{w}_j|, \alpha)$. In this case, (2) holds with $\mathbf{h}(\mathbf{w}) = [|\mathbf{w}_1|, \ldots, |\mathbf{w}_d|]$ and $g_\mathbf{h}^*(\mathbf{v}) = \sum_{j=1}^d \alpha(1 - \mathbf{v}_j)I(\mathbf{v}_j \in [0, 1])$ defined on the domain $\{\mathbf{v} : \mathbf{v}_j \geq 0\}$, where $I(\cdot)$ is the set indicator function. The solution in (4) is given by $\hat{\mathbf{v}}_j = I(|\mathbf{w}_j| \leq \alpha)$.

## 3   Multi-stage Convex Relaxation

We consider a general procedure for solving (2) with convex loss and non-convex regularization $g(\mathbf{w})$. Let $h(\mathbf{w}) = \sum_j \mathbf{h}_j(\mathbf{w})$ be a convex relaxation of $g(\mathbf{w})$ that dominates $g(\mathbf{w})$ (for example, it can be the smallest convex upperbound (i.e., the inf over all convex upperbounds) of $g(\mathbf{w})$). A simple convex relaxation of (2) becomes

$$\hat{\mathbf{w}} = \arg \min_{\mathbf{w} \in R^d} \left[ \frac{1}{n} \sum_{i=1}^n \phi(\mathbf{w}^T \mathbf{x}_i, y_i) + \lambda \sum_{j=1}^d \mathbf{h}_j(\mathbf{w}) \right]. \tag{5}$$

This simple relaxation can yield a solution that is not close to the solution of (2). However, if $\mathbf{h}$ satisfies the condition of Section 2, then it is possible to write $g(\mathbf{w})$ as (3). Now, with this new representation, we can rewrite (2) as

$$[\hat{\mathbf{w}}, \hat{\mathbf{v}}] = \arg \min_{\mathbf{w}, \mathbf{v} \in R^d} \left[ \frac{1}{n} \sum_{i=1}^n \phi(\mathbf{w}^T \mathbf{x}_i, y_i) + \lambda \mathbf{v}^T \mathbf{h}(\mathbf{w}) + \lambda g_\mathbf{h}^*(\mathbf{v}), \right], \tag{6}$$

This is clearly equivalent to (2) because of (3). If we can find a good approximation of $\hat{\mathbf{v}}$ that improves upon the initial value of $\hat{\mathbf{v}} = [1, \ldots, 1]$, then the above formulation can lead to a refined convex problem in $\mathbf{w}$ that is a better convex relaxation than (5).

Our numerical procedure exploits the above fact, which tries to improve the estimation of $\mathbf{v}_j$ over the initial choice of $\mathbf{v}_j = 1$ in (5) using an iterative algorithm. This can be done using an alternating optimization procedure, which repeatedly applies the following two steps:

- First we optimize $\mathbf{w}$ with $\mathbf{v}$ fixed: this is a convex problem in $\mathbf{w}$ with appropriately chosen $\mathbf{h}(\mathbf{w})$.
- Second we optimize $\mathbf{v}$ with $\mathbf{w}$ fixed: although non-convex, it has a closed form solution that is given by (4).

The general procedure is presented in Figure 1. It can be regarded as a generalization of CCCP (concave-convex programming) [7], which takes $\mathbf{h}(\mathbf{w}) = \mathbf{w}$. By repeatedly refining the parameter $\mathbf{v}$, we can potentially obtain better and better convex relaxation, leading to a solution superior to that of the initial convex relaxation. Note that using the $L_p$ and capped-$L_1$ regularization conditions in Section 2, this procedure lead to more specific multi-stage convex relaxation algorithms. We skip the details due to the space limitation.

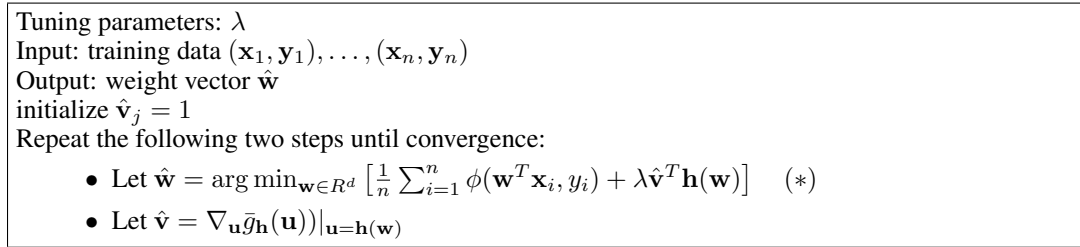

Figure 1: Multi-stage Convex Relaxation Method

## 4 Theory of Two-stage Convex Relaxation for Capped-$L_1$ Regularization

Although the reasoning in Section 3 is appealing, it is only a heuristic argument without any formal theoretical guarantee. In contrast, the simple one-stage $L_1$ relaxation is known to perform reasonably well under certain assumptions. Therefore unless we can develop a theory to show the effectiveness of the multi-stage procedure in Figure 1, our proposal is mere yet another local minimum finding scheme that may potentially stuck into a bad local solution.

This section tries to address this issue. Although we have not yet developed a complete theory for the general procedure, we are able to obtain a learning bound for the capped-$L_1$ regularization. In particular, if the target function is sparse, then the performance of the solution after merely two-stages of our procedure is superior to that of Lasso. This demonstrates the effectiveness of the multi-stage approach. Since the analysis is rather complicated, we focus on the least squares loss only, and only for the solution after two-stages of the algorithm.

For a complete theory, the following questions are worth asking:

- Under what conditions, the global solution with non-convex penalty is statistically better than the (one-stage) convex relaxation solution? That is, when does it lead to better prediction accuracy or generalization error?
- Under what conditions, there is only one local minimum solution close to the solution of the initial convex relaxation, and it is also the global optimum? Moreover, does multi-stage convex relaxation find this solution?

The first question answers whether it is beneficial to use a non-convex penalty function. The second question answers whether we can effectively solve the resulting non-convex problem using multi-stage convex relaxation. The combination of the two questions leads to a satisfactory theoretical answer to the effectiveness of the multi-stage procedure.

A general theory along this line will be developed in the full paper. In the following, instead of trying to answer the above questions separately, we provide a unified finite sample analysis for the procedure that directly addresses the combined effect of the two questions. The result is adopted

from [8], which justifies the multi-stage convex relaxation approach by showing that the two-stage procedure using capped-$L_1$ regularization can lead to better generalization than the standard one stage $L_1$ regularization.

The procedure we shall analyze, which is a special case of the multi-stage algorithm in Figure 1 with capped-$L_1$ regularization and only two stages, is described in Figure 2. It is related to the adaptive Lasso method [10]. The result is reproducible when the solution of the first stage is unique because it involves two well-defined convex programming problems. Note that it is described with least squares loss only because our analysis assumes least squares loss: a more general analysis for other loss functions is possible but would lead to extra complications that are not central to our interests.

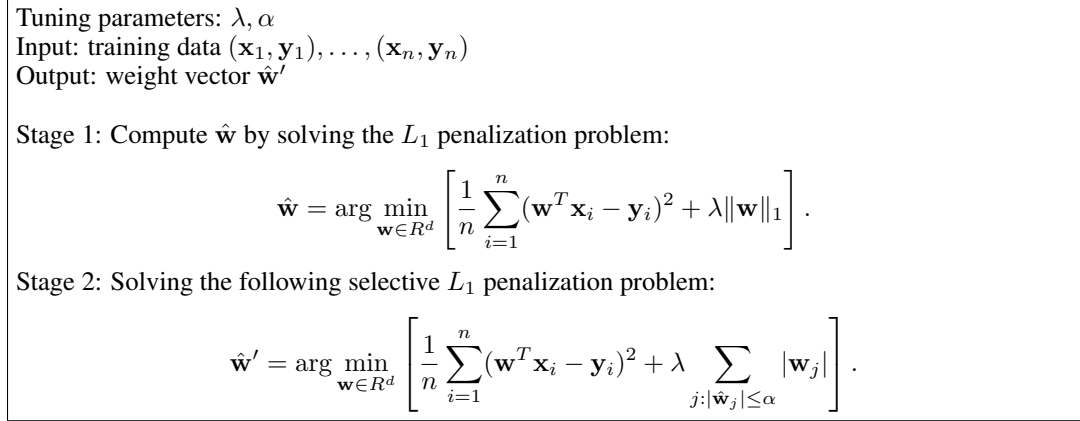

Tuning parameters: $\lambda, \alpha$
Input: training data $(\mathbf{x}_1, \mathbf{y}_1), \ldots, (\mathbf{x}_n, \mathbf{y}_n)$
Output: weight vector $\hat{\mathbf{w}}'$

Stage 1: Compute $\hat{\mathbf{w}}$ by solving the $L_1$ penalization problem:

$$\hat{\mathbf{w}} = \arg\min_{\mathbf{w} \in R^d} \left[ \frac{1}{n} \sum_{i=1}^{n} (\mathbf{w}^T \mathbf{x}_i - \mathbf{y}_i)^2 + \lambda \|\mathbf{w}\|_1 \right].$$

Stage 2: Solving the following selective $L_1$ penalization problem:

$$\hat{\mathbf{w}}' = \arg\min_{\mathbf{w} \in R^d} \left[ \frac{1}{n} \sum_{i=1}^{n} (\mathbf{w}^T \mathbf{x}_i - \mathbf{y}_i)^2 + \lambda \sum_{j : |\hat{\mathbf{w}}_j| \leq \alpha} |\mathbf{w}_j| \right].$$

Figure 2: Two-stage capped-$L_1$ Regularization

This particular two-stage procedure also has an intuitive interpretation (besides treating it as a special case of multi-stage convex relaxation). We shall refer to the feature components corresponding to the large weights as *relevant features*, and the feature components smaller the cut-off threshold $\alpha$ as *irrelevant features*. We observe that as an estimation method, $L_1$ regularization has two important properties: shrink estimated weights corresponding to irrelevant features toward zero; shrink estimated weights corresponding to relevant features toward zero. While the first effect is desirable, the second effect is not. In fact, we should avoid shrinking the weights corresponding to the relevant features if we can identify these features. This is why the standard $L_1$ regularization may have suboptimal performance. However, after the first stage of $L_1$ regularization, we can identify the relevant features by picking the components corresponding to the largest weights; in the second stage of $L_1$ regularization, we do not have to penalize the features selected in the first stage, as in Figure 2.

A related method, called *relaxed Lasso*, was proposed recently by Meinshausen [5], which is similar to a two-stage Dantzig selector in [2]. Their idea differs from our proposal in that in the second stage, the weight coefficients $\mathbf{w}'_j$ are forced to be zero when $j \notin \mathrm{supp}_0(\hat{\mathbf{w}})$. It was pointed out in [5] that if $\mathrm{supp}_0(\hat{\mathbf{w}})$ can exactly identify all non-zero components of the target vector, then in the second stage, the relaxed Lasso can asymptotically remove the bias in the first stage Lasso. However, it is not clear what theoretical result can be stated when Lasso cannot exactly identify all relevant features. In the general case, it is not easy to ensure that relaxed Lasso does not degrade the performance when some relevant coefficients become zero in the first stage. On the contrary, the two-stage penalization procedure in Figure 2, which is based on the capped-$L_1$ regularization, does not require that all relevant features are identified. Consequently, we are able to prove a result for Figure 2 with no counterpart for relaxed Lasso.

**Definition 4.1** *Let* $\mathbf{w} = [\mathbf{w}_1, \ldots, \mathbf{w}_d] \in R^d$ *and* $\alpha \geq 0$*, we define the set of relevant features with threshold* $\alpha$ *as:*

$$\mathrm{supp}_\alpha(\mathbf{w}) = \{j : |\mathbf{w}_j| > \alpha\}.$$

*Moreover, if* $|\mathbf{w}_{i_1}| \geq \cdots \geq |\mathbf{w}_{i_d}|$ *are in descending order, then define* $\delta_k(\mathbf{w}) = \left( \sum_{j>k} |\mathbf{w}_{i_j}|^2 \right)^{1/2}$ *as the 2-norm of the largest* $k$ *components (in absolute value) of* $\mathbf{w}$*.*

For simplicity, we assume sub-Gaussian noise as follows.

**Assumption 4.1** *Assume that* $\{\mathbf{y}_i\}_{i=1,\ldots,n}$ *are independent (but not necessarily identically distributed) sub-Gaussians: there exists* $\sigma \geq 0$ *such that* $\forall i$ *and* $\forall t \in R$,

$$\mathbf{E}_{\mathbf{y}_i} e^{t(\mathbf{y}_i - \mathbf{E}\mathbf{y}_i)} \leq e^{\sigma^2 t^2/2}.$$

Both Gaussian and bounded random variables are sub-Gaussian using the above definition. For example, if a random variable $\xi \in [a, b]$, then $\mathbf{E}_\xi e^{t(\xi - \mathbf{E}\xi)} \leq e^{(b-a)^2 t^2/8}$. If a random variable is Gaussian: $\xi \sim N(0, \sigma^2)$, then $\mathbf{E}_\xi e^{t\xi} \leq e^{\sigma^2 t^2/2}$.

**Theorem 4.1** *Let Assumption 4.1 hold. Let* $\hat{A} = \frac{1}{n} \sum_{i=1}^n \mathbf{x}_i \mathbf{x}_i^T$, *define* $M_{\hat{A}} = \sup_{i \neq j} |\hat{A}_{i,j}|$, *and assume that* $\hat{A}_{j,j} = 1$ *for all* $j$. *Consider any target vector* $\bar{\mathbf{w}}$ *such that* $\mathbf{E}y = \bar{\mathbf{w}}^T \mathbf{x}$, *and assume that* $\bar{\mathbf{w}}$ *contains only* $s$ *non-zeros where* $s \leq d/3$ *and assume that* $M_{\hat{A}} s \leq 1/6$. *Let* $k = |\mathrm{supp}_\lambda(\bar{\mathbf{w}})|$. *Consider the two-stage method in Figure 2. Given* $\eta \in (0, 0.5)$, *with probability larger than* $1 - 2\eta$: *if* $\alpha/48 \geq \lambda \geq 12\sigma\sqrt{2 \ln(2d/\eta)/n}$, *then*

$$\|\hat{\mathbf{w}}' - \bar{\mathbf{w}}\|_2 \leq 24\sqrt{k-q}\lambda + 24\sigma \left( 1 + \sqrt{\frac{20q}{n} \ln(1/\eta)} \right) + 168\delta_k(\bar{\mathbf{w}}),$$

*where* $q = |\mathrm{supp}_{1.5\alpha}(\bar{\mathbf{w}})|$.

The proof of this theorem can be found in [8]. Note that the theorem allows the situation $d \gg n$, which is what we are interested in. The condition $M_{\hat{A}} s \leq 1/6$, often referred to as *mutual coherence*, is also quite standard in the analysis of $L_1$ regularization, e.g., in [1, 3]. Although the condition is idealized, the theorem nevertheless yields important insights into the behavior of the two-stage algorithm. This theorem leads to a bound for Lasso with $\alpha = \infty$ or $q = 0$. The bound has the form

$$\|\hat{\mathbf{w}}' - \bar{\mathbf{w}}\|_2 = O(\delta_k(\bar{\mathbf{w}}) + \sqrt{k}\lambda).$$

This bound is tight for Lasso, in the sense that the right hand side cannot be improved except for the constant. In particular, the factor $O(\sqrt{k}\lambda)$ cannot be removed using Lasso — this can be easily verified with an orthogonal design matrix. It is known that in order for Lasso to be effective, one has to pick $\lambda$ no smaller than the order $\sigma\sqrt{\ln d/n}$. Therefore, the generalization of standard Lasso is of the order $\delta_k(\bar{\mathbf{w}}) + \sigma\sqrt{k \ln d/n}$, which cannot be improved. Similar results appear in [1, 4].

Now, with a small $\alpha$, the bound in Theorem 4.1 can be significantly better than that of the standard Lasso result if the sparse target satisfies $\delta_k(\bar{\mathbf{w}}) \ll \sqrt{k}\lambda$ and $k - q \ll k$. The latter condition is true when $|\mathrm{supp}_{1.5\alpha}(\bar{\mathbf{w}})| \approx |\mathrm{supp}_\lambda(\bar{\mathbf{w}})|$. These conditions are satisfied when most non-zero coefficients of $\bar{\mathbf{w}}$ in $\mathrm{supp}_\lambda(\bar{\mathbf{w}})$ are relatively large in magnitude and the rest is small in 2-norm. That is, when the target $\bar{\mathbf{w}}$ can be decompose as a sparse vector with large coefficients plus another (less sparse) vector with small coefficients. In the extreme case when $q = k = |\mathrm{supp}_0(\bar{\mathbf{w}})|$ (that is, all nonzero components of $\bar{\mathbf{w}}$ are large), we obtain $\|\hat{\mathbf{w}}' - \bar{\mathbf{w}}\|_2 = O(\sqrt{k \ln(1/\eta)/n})$ for the two-stage procedure, which is superior to the standard one-stage Lasso bound $\|\hat{\mathbf{w}} - \bar{\mathbf{w}}\|_2 = O(\sqrt{k \ln(d/\eta)/n})$. Again, this bound cannot be improved for Lasso, and the difference can be significant when $d$ is large.

## 5  Experiments

In the following, we show with a synthetic and a real data that our multi-stage approach improves the standard Lasso in practice. In order to avoid cluttering, we only study results for the two-stage procedure of Figure 2, which corresponds to the capped-$L_1$ regularization. We shall also compare it to the two-stage $L_p$ regularization method with $p = 0.5$, which corresponds to the adaptive Lasso approach [10]. Note that instead of tuning the $\alpha$ parameter in Figure 2, in these experiments, we tune the number of features $q$ in $\hat{\mathbf{w}}$ that are larger than the threshold $\alpha$ (i.e., $q = |\{j : |\hat{\mathbf{w}}_j| > \alpha\}|$ is the number of features that are not regularized in stage-2). This is clearly more convenient than tuning $\alpha$. The standard Lasso corresponds to $q = 0$.

In the first experiment, we generate an $n \times d$ random matrix with its column $j$ corresponding to $[\mathbf{x}_{1,j}, \ldots, \mathbf{x}_{n,j}]$, and each element of the matrix is an independent standard Gaussian $N(0, 1)$. We then normalize its columns so that $\sum_{i=1}^n \mathbf{x}_{i,j}^2 = n$. A truly sparse target $\bar{\beta}$, is generated with $k$

nonzero elements that are uniformly distributed from $[-10, 10]$. The observation $\mathbf{y}_i = \bar{\beta}^T \mathbf{x}_i + \epsilon_i$, where each $\epsilon_i \sim N(0, \sigma^2)$. In this experiment, we take $n = 25, d = 100, k = 5, \sigma = 1$, and repeat the experiment 100 times. The average training error and 2-norm parameter estimation error are reported in Figure 3. We compare the performance of the two-stage method with different $q$ versus the regularization parameter $\lambda$. As expected, the training error becomes smaller when $q$ increases. Compared to the standard Lasso (which corresponds to $q = 0$), substantially smaller estimation error is achieved with $q = 3$ for Capped-$L_1$ regularization and with $p = 0.5$ for $L_p$ regularization. This shows that the multi-stage convex relaxation approach is effective.

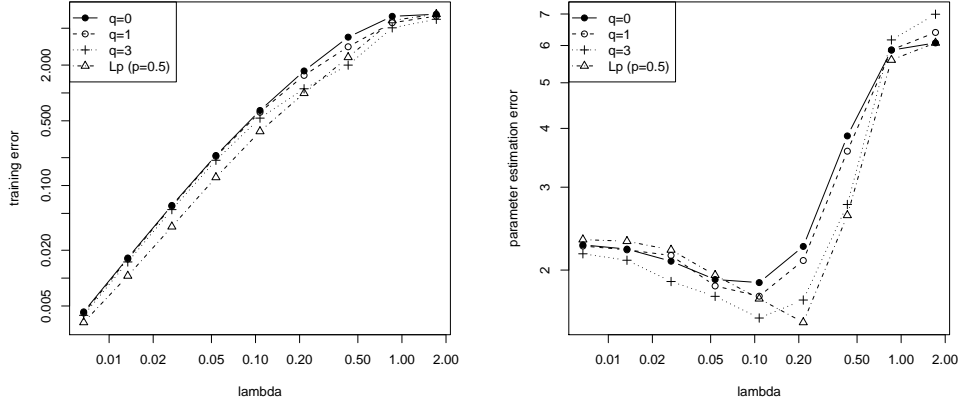

Figure 3: Performance of multi-stage convex relaxation on simulation data. Left: average training squared error versus $\lambda$; Right: parameter estimation error versus $\lambda$.

In the second experiment, we use real data to illustrate the effectiveness of the multi-stage approach. Due to the space limitation, we only report the performance on a single data, *Boston Housing*. This is the housing data for 506 census tracts of Boston from the 1970 census, available from the *UCI Machine Learning Database Repository*: *http://archive.ics.uci.edu/ml/*. Each census tract is a data-point, with 13 features (we add a constant offset on e as the 14th feature), and the desired output is the housing price. In the experiment, we randomly partition the data into 20 training plus 456 test points. We perform the experiments 100 times, and report training and test squared error versus the regularization parameter $\lambda$ for different $q$. The results are plotted in Figure 4. In this case, $q = 1$ achieves the best performance. This means one feature can be reliably identified in this example. In comparison, adaptive Lasso is not effective. Note that this dataset contains only a small number ($d = 14$) features, which is not the case where we can expect significant benefit from the multi-stage approach (most of other UCI data similarly contain only small number of features). In order to illustrate the advantage of the two-stage method more clearly, we also consider a modified Boston Housing data, where we append 20 random features (similar to the simulation experiments) to the original Boston Housing data, and rerun the experiments. The results are shown in Figure 5. As expected from Theorem 4.1 and the discussion thereafter, since $d$ becomes large, the multi-stage convex relaxation approach with capped-$L_1$ regularization ($q > 0$) has significant advantage over the standard Lasso ($q = 0$).

# References

[1] Florentina Bunea, Alexandre Tsybakov, and Marten H. Wegkamp. Sparsity oracle inequalities for the Lasso. *Electronic Journal of Statistics*, 1:169–194, 2007.

[2] Emmanuel Candes and Terence Tao. The Dantzig selector: statistical estimation when $p$ is much larger than $n$. *Annals of Statistics*, 2007.

[3] David L. Donoho, Michael Elad, and Vladimir N. Temlyakov. Stable recovery of sparse over-complete representations in the presence of noise. *IEEE Trans. Info. Theory*, 52(1):6–18, 2006.

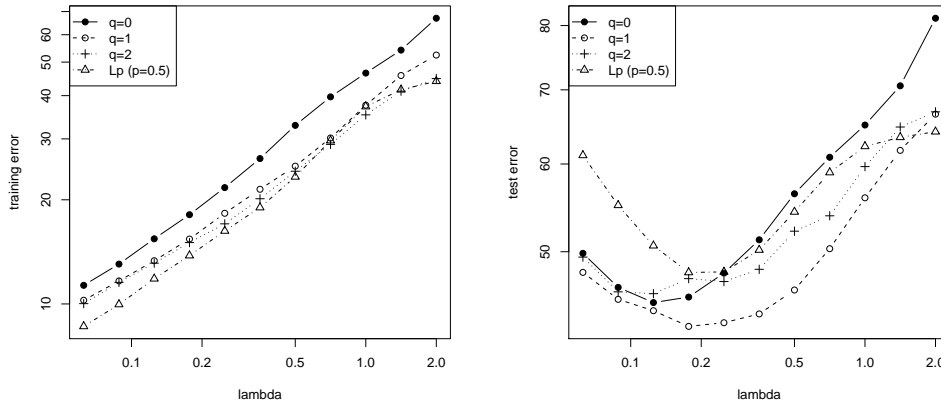

Figure 4: Performance of multi-stage convex relaxation on the original Boston Housing data. Left: average training squared error versus $\lambda$; Right: test squared error versus $\lambda$.

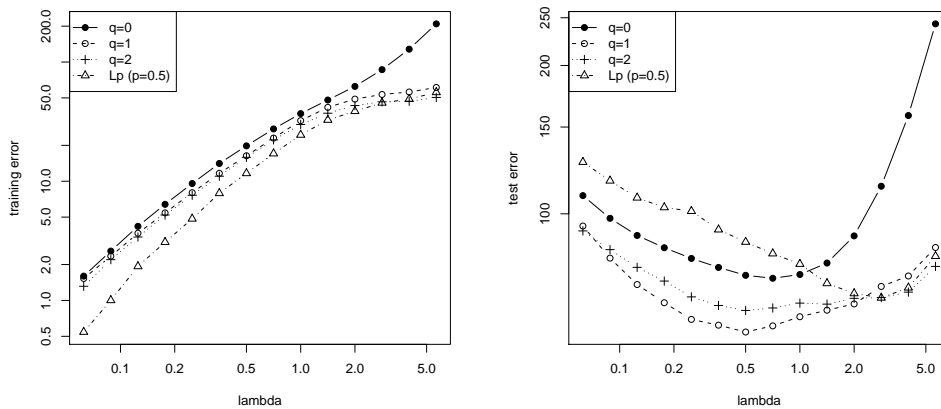

Figure 5: Performance of multi-stage convex relaxation on the modified Boston Housing data. Left: average training squared error versus $\lambda$; Right: test squared error versus $\lambda$.

[4] Vladimir Koltchinskii. Sparsity in penalized empirical risk minimization. *Annales de l'Institut Henri Poincaré*, 2008.

[5] Nicolai Meinshausen. Lasso with relaxation. ETH Research Report, 2005.

[6] R. Tyrrell Rockafellar. *Convex analysis*. Princeton University Press, Princeton, NJ, 1970.

[7] Alan L. Yuille and Anand Rangarajan. The concave-convex procedure. *Neural Computation*, 15:915–936, 2003.

[8] Tong Zhang. Some sharp performance bounds for least squares regression with $L_1$ regularization. *The Annals of Statistics*, 2009. to appear.

[9] Peng Zhao and Bin Yu. On model selection consistency of Lasso. *Journal of Machine Learning Research*, 7:2541–2567, 2006.

[10] Hui Zou. The adaptive lasso and its oracle properties. *Journal of the American Statistical Association*, 101:1418–1429, 2006.

